# Gaussian Process Training with Input Noise

**Andrew McHutchon**
Department of Engineering
Cambridge University
Cambridge, CB2 1PZ
ajm257@cam.ac.uk

**Carl Edward Rasmussen**
Department of Engineering
Cambridge University
Cambridge, CB2 1PZ
cer54@cam.ac.uk

## Abstract

In standard Gaussian Process regression input locations are assumed to be noise free. We present a simple yet effective GP model for training on input points corrupted by i.i.d. Gaussian noise. To make computations tractable we use a local linear expansion about each input point. This allows the input noise to be recast as output noise proportional to the squared gradient of the GP posterior mean. The input noise variances are inferred from the data as extra hyperparameters. They are trained alongside other hyperparameters by the usual method of maximisation of the marginal likelihood. Training uses an iterative scheme, which alternates between optimising the hyperparameters and calculating the posterior gradient. Analytic predictive moments can then be found for Gaussian distributed test points. We compare our model to others over a range of different regression problems and show that it improves over current methods.

## 1 Introduction

Over the last decade the use of Gaussian Processes (GPs) as non-parametric regression models has grown significantly. They have been successfully used to learn mappings between inputs and outputs in a wide variety of tasks. However, many authors have highlighted a limitation in the way GPs handle noisy measurements. Standard GP regression [1] makes two assumptions about the noise in datasets: firstly that measurements of input points, $x$, are noise-free, and, secondly, that output points, $y$, are corrupted by constant-variance Gaussian noise. For some datasets this makes intuitive sense: for example, an application in Rasmussen and Williams (2006) [1] is that of modelling $CO_2$ concentration in the atmosphere over the last forty years. One can viably assume that the date is available noise-free and the $CO_2$ sensors are affected by signal-independent sensor noise.

However, in many datasets, either or both of these assumptions are not valid and lead to poor modelling performance. In this paper we look at datasets where the input measurements, as well as the output, are corrupted by noise. Unfortunately, in the GP framework, considering each input location to be a distribution is intractable. If, as an approximation, we treat the input measurements as if they were deterministic, and inflate the corresponding output variance to compensate, this leads to the output noise variance varying across the input space, a feature often called *heteroscedasticity*. One method for modelling datasets with input noise is, therefore, to hold the input measurements to be deterministic and then use a heteroscedastic GP model. This approach has been strengthened by the breadth of research published recently on extending GPs to heteroscedastic data.

However, referring the input noise to the output in this way results in heteroscedasticity with a very particular structure. This structure can be exploited to improve upon current heteroscedastic GP models for datasets with input noise. One can imagine that in regions where a process is changing its output value rapidly, corrupted input measurements will have a much greater effect than in regions

*Pre-conference version*

where the output is almost constant. In other words, the effect of the input noise is related to the gradient of the function mapping input to output. This is the intuition behind the model we propose in this paper.

We fit a local linear model to the GP posterior mean about each training point. The input noise variance can then be referred to the output, proportional to the square of the posterior mean function's gradient. This approach is particularly powerful in the case of time-series data where the output at time $t$ becomes the input at time $t + 1$. In this situation, input measurements are clearly not noise-free: the noise on a particular measurement is the same whether it is considered an input or output. By also assuming the inputs are noisy, our model is better able to fit datasets of this type. Furthermore, we can estimate the noise variance on each input dimension, which is often very useful for analysis.

Related work lies in the field of heteroscedastic GPs. A common approach to modelling changing variance with a GP, as proposed by Goldberg et al. [2], is to make the noise variance a random variable and attempt to estimate its form at the same time as estimating the posterior mean. Goldberg et al. suggested using a second GP to model the noise level as a function of the input location. Kersting et al. [3] improved upon Goldberg et al.'s Monte Carlo training method with a "most likely" training scheme and demonstrated its effectiveness; related work includes Yuan and Wahba [4], and Le at al. [5] who proposed a scheme to find the variance via a maximum-a-posteriori estimate set in the exponential family. Snelson and Ghahramani [6] suggest a different approach whereby the importance of points in a pseudo-training set can be varied, allowing the posterior variance to vary as well. Recently Wilson and Ghahramani broadened the scope still further and proposed Copula and Wishart Process methods [7, 8].

Although all of these methods could be applied to datasets with input noise, they are designed for a more general class of heteroscedastic problems and so none of them exploits the structure inherent in input noise datasets. Our model also has a further advantage in that training is by marginal likelihood maximisation rather than by an approximate inference method, or one such as maximum likelihood, which is more susceptible to overfitting. Dallaire et al. [9] train on Gaussian distributed input points by calculating the expected the covariance matrix. However, their method requires prior knowledge of the noise variance, rather than inferring it as we do in this paper.

## 2   The Model

In this section we formally derive our model, which we refer to as NIGP (noisy input GP).

Let $\boldsymbol{x}$ and $y$ be a pair of measurements from a process, where $\boldsymbol{x}$ is a $D$ dimensional input to the process and $y$ is the corresponding scalar output. In standard GP regression we assume that $y$ is a noisy measurement of the actual output of the process $\tilde{y}$,

$$y = \tilde{y} + \epsilon_y \tag{1}$$

where, $\epsilon_y \sim \mathcal{N}\left(0, \sigma_y^2\right)$. In our model, we further assume that the inputs are also noisy measurements of the actual input $\tilde{\boldsymbol{x}}$,

$$\boldsymbol{x} = \tilde{\boldsymbol{x}} + \boldsymbol{\epsilon}_x \tag{2}$$

where $\boldsymbol{\epsilon}_x \sim \mathcal{N}\left(\boldsymbol{0}, \Sigma_x\right)$. We assume that each input dimension is independently corrupted by noise, thus $\Sigma_x$ is diagonal. Under a model $f(.)$, we can write the output as a function of the input in the following form,

$$y = f(\tilde{\boldsymbol{x}} + \boldsymbol{\epsilon}_x) + \epsilon_y \tag{3}$$

For a GP model the posterior distribution based on equation 3 is intractable. We therefore consider a Taylor expansion about the latent state $\tilde{\boldsymbol{x}}$,

$$f(\tilde{\boldsymbol{x}} + \boldsymbol{\epsilon}_x) \;=\; f(\tilde{\boldsymbol{x}}) + \boldsymbol{\epsilon}_x^T \frac{\partial f(\tilde{\boldsymbol{x}})}{\partial \tilde{\boldsymbol{x}}} + \ldots \;\simeq\; f(\boldsymbol{x}) + \boldsymbol{\epsilon}_x^T \frac{\partial f(\boldsymbol{x})}{\partial \boldsymbol{x}} + \ldots \tag{4}$$

We don't have access to the latent variable $\tilde{\boldsymbol{x}}$ so we approximate it with the noisy measurements. Now the derivative of a Gaussian Process is another Gaussian Process [10]. Thus, the exact treatment would require the consideration of a distribution over Taylor expansions. Although the resulting distribution is not Gaussian, its first and second moments can be calculated analytically. However, these calculations carry a high computational load and previous experiments showed this exact treatment

provided no significant improvement over the much quicker approximate method we now describe. Instead we take the derivative of the mean of the GP function, which we will denote $\boldsymbol{\partial}_{\bar{f}}$, a $D$-dimensional vector, for the derivative of one GP function value w.r.t. the $D$-dimensional input, and $\Delta_{\bar{f}}$, an $N$ by $D$ matrix, for the derivative of $N$ function values. Differentiating the mean function corresponds to ignoring the uncertainty about the derivative. If we expand up to the first order terms we get a linear model for the input noise,

$$y = f(\boldsymbol{x}) + \boldsymbol{\epsilon}_x^T \boldsymbol{\partial}_{\bar{f}} + \epsilon_y \tag{5}$$

The probability of an observation $y$ is therefore,

$$P(y \mid f) \;=\; \mathcal{N}(f, \; \sigma_y^2 + \boldsymbol{\partial}_{\bar{f}}^T \Sigma_x \, \boldsymbol{\partial}_{\bar{f}}) \tag{6}$$

We keep the usual Gaussian Process prior, $P(\boldsymbol{f} \mid X) = \mathcal{N}(\boldsymbol{0}, K(X,X))$, where $K(X,X)$ is the $N$ by $N$ training data covariance matrix and $X$ is an $N$ by $D$ matrix of input observations. Combining these probabilities gives the predictive posterior mean and variance as,

$$\mathbb{E}\left[f_* \mid X, \boldsymbol{y}, \boldsymbol{x}_*\right] \;=\; \boldsymbol{k}(\boldsymbol{x}_*, X)\big[K(X,X) + \sigma_y^2 I + \mathrm{diag}\{\Delta_{\bar{f}} \, \Sigma_x \, \Delta_{\bar{f}}^T\}\big]^{-1} \boldsymbol{y}$$
$$\mathbb{V}\left[f_* \mid X, \boldsymbol{y}, \boldsymbol{x}_*\right] \;=\; k(\boldsymbol{x}_*, \boldsymbol{x}_*) \;-\; \boldsymbol{k}(\boldsymbol{x}_*, X)\big[K(X,X) + \sigma_y^2 I + \mathrm{diag}\{\Delta_{\bar{f}} \, \Sigma_x \, \Delta_{\bar{f}}^T\}\big]^{-1} \boldsymbol{k}(X, \boldsymbol{x}_*) \tag{7}$$

This is equivalent to treating the inputs as deterministic and adding a corrective term, $\mathrm{diag}\{\Delta_{\bar{f}} \, \Sigma_x \, \Delta_{\bar{f}}^T\}$, to the output noise. The notation "diag{.}" results in a diagonal matrix, the elements of which are the diagonal elements of its matrix argument. Note that if the posterior mean gradient is constant across the input space the heteroscedasticity is removed and our model is essentially identical to a standard GP.

An advantage of our approach can be seen in the case of multiple output dimensions. As the input noise levels are the same for each of the output dimensions, our model can use data from all of the outputs when learning the input noise variances. Not only does this give more information about the noise variances without needing further input measurements but it also reduces over-fitting as the learnt noise variances must agree with all $E$ output dimensions.

For time-series datasets (where the model has to predict the next state given the current), each dimension's input and output noise variance can be constrained to be the same since the noise level on a measurement is independent of whether it is an input or output. This further constraint increases the ability of the model to recover the actual noise variances. The model is thus ideally suited to the common task of multivariate time series modelling.

## 3  Training

Our model introduces an extra $D$ hyperparameters compared to the standard GP - one noise variance hyperparameter per input dimension. A major advantage of our model is that these hyperparameters can be trained alongside any others by maximisation of the marginal likelihood. This approach automatically includes regularisation of the noise parameters and reduces the effect of over-fitting.

In order to calculate the marginal likelihood of the training data we need the posterior distribution, and the slope of its mean, at each of the training points. However, evaluating the posterior mean from equation 7 with $\boldsymbol{x}_* \in X$, results in an analytically unsolvable differential equation: $\bar{f}$ is a complicated function of $\Delta_{\bar{f}}$, its own derivative. Therefore, we define a two-step approach: first we evaluate a standard GP with the training data, using our initial hyperparameter settings and ignoring the input noise. We then find the slope of the posterior mean of this GP at each of the training points and use it to add in the corrective variance term, $\mathrm{diag}\{\Delta_{\bar{f}} \, \Sigma_x \, \Delta_{\bar{f}}^T\}$. This process is summarised in figures 1a and 1b.

The marginal likelihood of the GP with the corrected variance is then computed, along with its derivatives with respect to the initial hyperparameters, which include the input noise variances. This step involves chaining the derivatives of the marginal likelihood back through the slope calculation. Gradient descent can then be used to improve the hyperparameters. Figure 1c shows the GP posterior for the trained hyperparameters and shows how NIGP can reduce output noise level estimates by taking input noise into account. Figure 1d shows the NIGP fit for the trained hyperparameters.

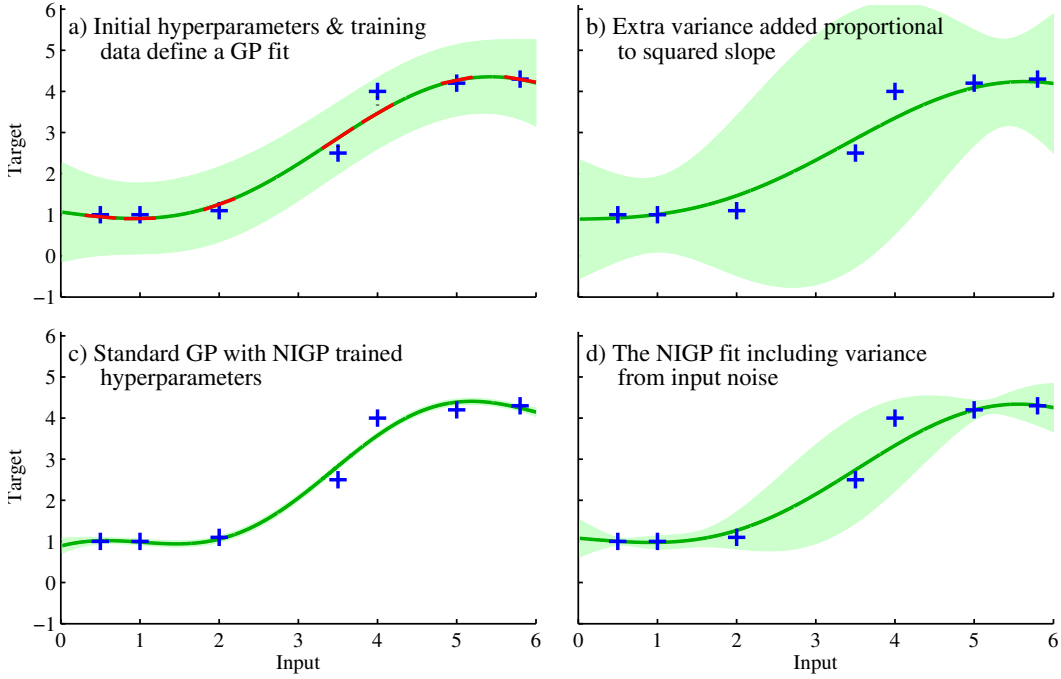

Figure 1: Training with NIGP. **(a)** A standard GP posterior distribution can be computed from an initial set of hyperparameters and a training data set, shown by the blue crosses. The gradients of the posterior mean at each training point can then be found analytically. **(b)** The NIGP method increases the posterior variance by the square of the posterior mean slope multiplied by the current setting of the input noise variance hyperparameter. The marginal likelihood of this fit is then calculated along with its derivatives w.r.t. initial hyperparameter settings. Gradient descent is used to train the hyperparameters. **(c)** This plot shows the standard GP posterior using the newly trained hyperparameters. Comparing to plot (a) shows that the output noise hyperparameter has been greatly reduced. **(d)** This plot shows the NIGP fit - plot(c) with the input noise corrective variance term, $\mathrm{diag}\{\Delta_{\bar{f}}\, \Sigma_x\, \Delta_{\bar{f}}^T\}$. Plot (d) is related to plot (c) in the same way that plot (b) is related to plot (a).

To improve the fit further we can iterate this procedure: we use the slopes of the current trained NIGP, instead of a standard GP, to calculate the effect of the input noise, i.e. replace the fit in figure 1a with the fit from figure 1d and re-train.

## 4 Prediction

We turn now to the task of making predictions at noisy input locations with our model. To be true to our model we must use the same process in making predictions as we did in training. We therefore use the trained hyperparameters and the training data to define a GP posterior mean, which we differentiate at each test point and each training point. The calculated gradients are then used to add in the corrective variance terms. The posterior mean slope at the test points is only used to calculate the variance over observations, where we increase the predictive variance by the noise variances.

There is an alternative option, however. If a single test point is considered to have a Gaussian distribution and all the training points are certain then, although the GP posterior is unknown, its mean and variance can be calculated exactly [11]. As our model estimates the input noise variance $\Sigma_x$ during training, we can consider a test point to be Gaussian distributed: $\boldsymbol{x}'_* \sim \mathcal{N}\left(\boldsymbol{x}_*, \Sigma_x\right)$. [11] then gives the mean and variance of the posterior distribution, for a squared exponential kernel (equation 12), to be,

$$\bar{f}_* = \left(\left[K + \sigma_y^2 I + \Sigma_x \boldsymbol{\partial}\bar{\boldsymbol{f}}^2\right]^{-1}\boldsymbol{y}\right)^T \boldsymbol{q} \tag{8}$$

where,

$$q_i = \sigma_f^2 \left| \Sigma_x \Lambda^{-1} + I \right|^{-\frac{1}{2}} \exp\left( -\frac{1}{2}(\boldsymbol{x}_i - \boldsymbol{x}_*)^T (\Sigma_x + \Lambda)^{-1} (\boldsymbol{x}_i - \boldsymbol{x}_*) \right) \tag{9}$$

where $\Lambda$ is a diagonal matrix of the squared lengthscale hyperparameters.

$$\mathbb{V}[f_*] = \sigma_f^2 - \operatorname{tr}\left( \left[ K + \sigma_y^2 I + \Sigma_x \boldsymbol{\partial} \bar{\boldsymbol{f}}^2 \right]^{-1} Q \right) + \boldsymbol{\alpha}^T Q \boldsymbol{\alpha} - \bar{f}_*^2 \tag{10}$$

with,

$$Q_{ij} = \frac{k(\boldsymbol{x}_i, \boldsymbol{x}_*) k(\boldsymbol{x}_j, \boldsymbol{x}_*)}{|2\Sigma_x \Lambda^{-1} + I|^{\frac{1}{2}}} \exp\left( (\boldsymbol{z} - \boldsymbol{x}_*)^T \left( \Lambda + \frac{1}{2}\Lambda \Sigma_x^{-1} \Lambda \right)^{-1} (\boldsymbol{z} - \boldsymbol{x}_*) \right) \tag{11}$$

with $\boldsymbol{z} = \frac{1}{2}(\boldsymbol{x}_i + \boldsymbol{x}_j)$. This method is computationally slower than using equation 7 and is vulnerable to worse results if the learnt input noise variance $\Sigma_x$ is very different from the true value. However, it gives proper consideration to the uncertainty surrounding the test point and exactly computes the moments of the correct posterior distribution. This often leads it to outperform predictions based on equation 7.

## 5  Results

We tested our model on a variety of functions and datasets, comparing its performance to standard GP regression as well as Kersting et al.'s 'most likely heteroscedastic GP' (MLHGP) model, a state-of-the-art heteroscedastic GP model. We used the squared exponential kernel with Automatic Relevance Determination,

$$k(\boldsymbol{x}_i, \boldsymbol{x}_j) = \sigma_f^2 \exp\left( -\frac{1}{2}(\boldsymbol{x}_i - \boldsymbol{x}_j)^T \Lambda^{-1} (\boldsymbol{x}_i - \boldsymbol{x}_j) \right) \tag{12}$$

where $\Lambda$ is a diagonal matrix of the squared lengthscale hyperparameters and $\sigma_f^2$ is a signal variance hyperparameter. Code to run NIGP is available on the author's website.

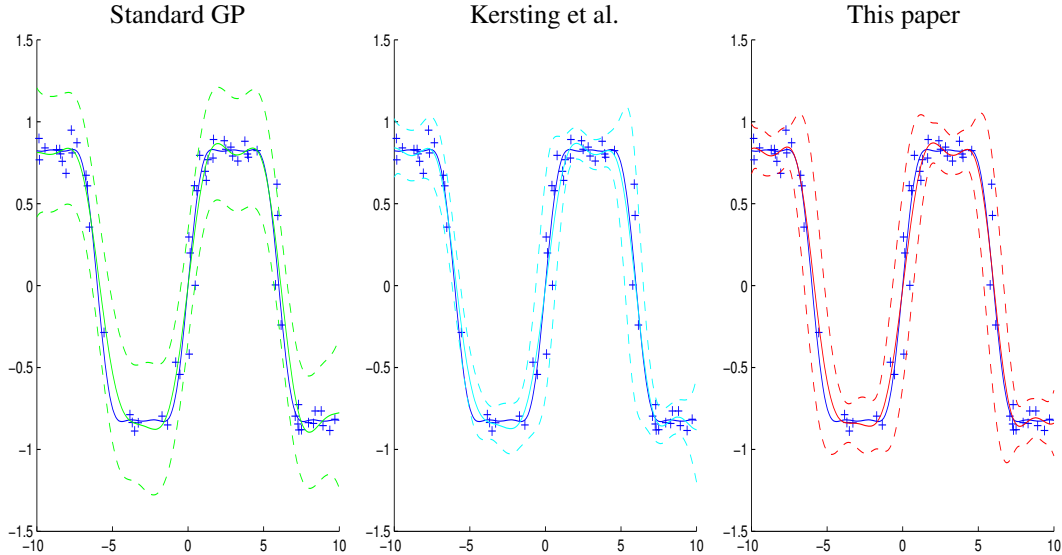

Figure 2: Posterior distribution for a near-square wave with $\sigma_y = 0.05$, $\sigma_x = 0.3$, and 60 data points. The solid line represents the predictive mean and the dashed lines are two standard deviations either side. Also shown are the training points and the underlying function. The left image is for standard GP regression, the middle uses Kersting et al.'s MLHGP algorithm, the right image shows our model. While the predictive means are similar, both our model and MLHGP pinch in the variance around the low noise areas. Our model correctly expands the variance around all steep areas whereas MLHGP can only do so where high noise is observed (see areas around x= -6 and x = 1).

Figure 2 shows an example comparison between standard GP regression, Kersting et al.'s MLHGP, and our model for a simple near-square wave function. This function was chosen as it has areas

of steep gradient and near flat gradient and thus suffers from the heteroscedastic problems we are trying to solve. The posterior means are very similar for the three models, however the variances are quite different. The standard GP model has to take into account the large noise seen around the steep sloped areas by assuming large noise everywhere, which leads to the much larger error bars. Our model can recover the actual noise levels by taking the input noise into account. Both our model and MLHGP pinch the variance in around the flat regions of the function and expand it around the steep areas. For the example shown in figure 2 the standard GP estimated an output noise standard deviation of 0.16 (much too large) compared to our estimate of 0.052, which is very close to the correct value of 0.050. Our model also learnt an input noise standard deviation of 0.305, very close to the real value of 0.300. MLHGP does not produce a single estimate of noise levels.

Predictions for 1000 noisy measurements were made using each of the models and the log probability of the test set was calculated. The standard GP model had a log probability per data point of 0.419, MLHGP 0.740, and our model 0.885, a significant improvement. Part of the reason for our improvement over MLHGP can be seen around $x = 1$: our model has near-symmetric 'horns' in the variance around the corners of the square wave, whereas MLHGP only has one 'horn'. This is because in our model, the amount of noise expected is proportional to the derivative of the mean squared, which is the same for both sides of the square wave. In Kersting et al.'s model the noise is estimated from the training points themselves. In this example the training points around $x = 1$ happen to have low noise and so the learnt variance is smaller. The same problem can be seen around $x = -6$ where MLHGP has much too small variance. This illustrates an important aspect of our model: the accuracy in plotting the varying effect of noise is only dependent on the accuracy of the mean posterior function and not on an extra, learnt noise model. This means that our model typically requires fewer data points to achieve the same accuracy as MLHGP on input noise datasets. To test the models further, we trained them on a suite of six functions. The functions were again chosen to have varying gradients across the input space. The training set consisted of twenty five points in the interval [-10, 10] and the test set one thousand points in the same interval. Trials were run for different levels of input noise. For each trial, ten different initialisations of the hyperparameters were tried. In order to remove initialisation effects the best initialisations for each model were chosen at each step. The entire experiment was run on twenty different random seeds. For our model, NIGP, we trained both a single model for all output dimensions, as well as separate models for each of the outputs, to see what the effect of using the cross-dimension information was.

Figure 3 shows the results for this experiment. The figure shows that NIGP performs very well on all the functions, always outperforming the standard GP when there is input noise and nearly always MLHGP; wherever there is a significant difference our model is favoured. Training on all the outputs at once only gives an improvement for some of the functions, which suggests that, for the others, the input noise levels could be estimated from the individual functions alone. The predictions using stochastic test-points, equations 8 and 10, generally outperformed the predictions made using deterministic test-points, equation 7. The RMSEs are quite similar to each other for most of the functions as the posterior means are very similar, although where they do differ significantly, again, it is to favour our model. These results show our model consistently calculates a more accurate predictive posterior variance than either a standard GP or a state-of-the-art heteroscedastic GP model.

As previously mentioned, our model can be adapted to work more effectively with time-series data, where the outputs become subsequent inputs. In this situation the input and output noise variance will be the same. We therefore combine these two parameters into one. We tested NIGP on a time-series dataset and compared the two modes (with separate input and output noise hyperparameters and with combined) and also to standard GP regression (MLHGP was not available for multiple input dimensions). The dataset is a simulated pendulum without friction and with added noise. There are two variables: pendulum angle and angular velocity. The choice of time interval between observations is important: for very small time intervals, and hence small changes in the angle, the dynamics are approximately linear, as $\sin \theta \approx \theta$. As discussed before, our model will not bring any benefit to linear dynamics, so in order to see the difference in performance a much longer time interval was chosen. The range of initial angular velocities was chosen to allow the pendulum to spin multiple times at the extremes, which adds extra non-linearity. Ten different initialisations were tried, with the one achieving the highest training set marginal likelihood chosen, and the whole experiment was repeated fifty times with different random seeds.

The plots show the difference in log probability of the test set between four versions of NIGP and a standard GP model trained on the same data. All four versions of our model perform better than the

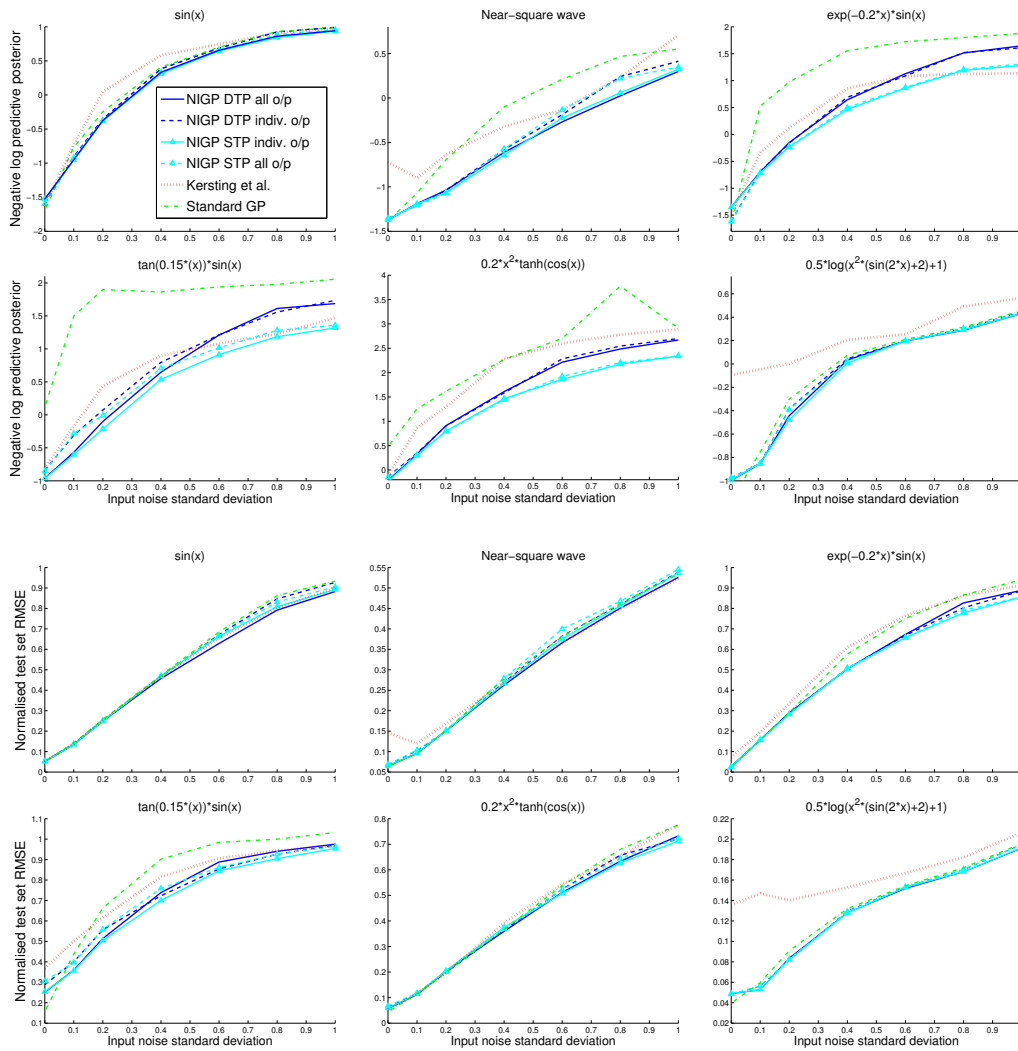

Figure 3: Comparison of models for suite of 6 test functions. The solid line is our model with 'deterministic test-point' predictions, the solid line with triangles is our model with 'stochastic test-point' predictions. Both these models were trained on all 6 functions at once, the respective dashed lines were trained on the functions individually. The dash-dot line is a standard GP regression model and the dotted line is MLHGP. RMSE has been normalised by the RMS value of the function. In both plots lower values indicate better performance. The plots show our model has lower negative log posterior predictive than standard GP on all the functions, particularly the exponentially decaying sine wave and the multiplication between tan and sin.

standard GP. Once again the stochastic test point version outperforms the deterministic test points. There was a slight improvement in RMSE using our model but the differences were within two standard deviations of each other. There is also a slight improvement using the combined noise levels although, again, the difference is contained within the error bars.

A better comparison between the two modes is to look at the input noise variance values recovered. The real noise standard deviations used were 0.2 and 0.4 for the angle and angular velocity respectively. The model which learnt the variances separately found standard deviations of 0.3265 and 0.8026 averaged over the trials, whereas the combined model found 0.2429 and 0.8948. This is a significant improvement on the first dimension. Both modes struggle to recover the correct noise level on the second dimension and this is probably why the angular velocity prediction performance shown in figure 4 is worse than the angle prediction performance. Training with more data signif-

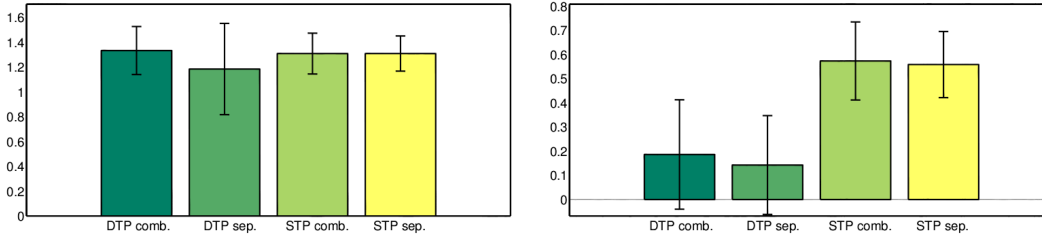

Figure 4: The difference between four versions of NIGP and a standard GP model on a pendulum prediction task. DTP stands for deterministic test point and STP is stochastic test point. Comb. and sep. indicate whether the model combined the input and output noise parameters or treated them separately. The error bars indicate plus/minus two standard deviations.

icantly improved the recovered noise value although the difference between the two NIGP modes then shrank as there was sufficient information to correctly deduce the noise levels separately.

## 6   Conclusion

The correct way of training on input points corrupted by Gaussian noise is to consider every input point as a Gaussian distribution. This model is intractable, however, and so approximations must be made. In our model, we refer the input noise to the output by passing it through a local linear expansion. This adds a term to the likelihood which is proportional to the squared posterior mean gradient. Not only does this lead to tractable computations but it makes intuitive sense - input noise has a larger effect in areas where the function is changing its output rapidly. The model, although simple in its approach, has been shown to be very effective, outperforming Kersting et al.'s model and a standard GP model in a variety of different regression tasks. It can make use of multiple outputs and can recover a noise variance parameter for each input dimension, which is often useful for analysis. In our approximate model, exact inference can be performed as the model hyperparameters can be trained simultaneously by marginal likelihood maximisation.

A proper handling of time-series data would constrain the specific noise levels on each training point to be the same for when they are considered inputs and outputs. This would be computationally very expensive however. By allowing input noise and fixing the input and output noise variances to be identical, our model is a computationally efficient alternative. Our results showed that NIGP gives a substantial improvement over the often-used standard GP for modelling time-series data.

It is important to state that this model has been designed to tackle a particular situation, that of constant-variance input noise, and would not perform so well on a general heteroscedastic problem. It could not be expected to improve over a standard GP on problems where noise levels are proportional to the function or input value for example. We do not see this limitation as too restricting however, as we maintain that constant input noise situations (including those where this is a sufficient approximation) are reasonably common. Throughout the paper we have taken particular care to avoid functions or systems which are linear, or approximately linear, as in these cases our model can be reduced to standard GP regression. However, for the problems for which NIGP has been designed, such as the various non-linear problems we have presented in this paper, our model outperforms current methods.

This paper considers a first order Taylor expansion of the posterior mean function. We would expect this to be a good approximation for any function providing the input noise levels are not too large (i.e. small perturbations around the point we linearised about). In practice, we could require that the input noise level is not larger than the input characteristic length scale. A more accurate model could use a second order Taylor series, which would still be analytic although computationally the algorithm would then scale with $D^3$ rather than the current $D^2$. Another refinement could be achieved by doing a Taylor series for the full posterior distribution (not just its mean, as we have done here), again at considerably higher computational cost. These are interesting areas for future research, which we are actively pursuing.

# References

[1] Carl Edward Rasmussen and Christopher K. I. Williams. *Gaussian Processes for Machine Learning*. MIT Press, 2006.

[2] Paul W. Goldberg, Christopher K. I. Williams, and Christopher M. Bishop. Regression with input-dependent noise: A Gaussian Process treatment. *NIPS-98*, 1998.

[3] Kristian Kersting, Christian Plagemann, Patrick Pfaff, and Wolfram Burgard. Most likely heteroscedastic Gaussian Process regression. *ICML-07*, 2007.

[4] Ming Yuan and Grace Wahba. Doubly penalized likelihood estimator in heteroscedastic regression. *Statistics and Probability Letter*, 69:11–20, 2004.

[5] Quoc V. Le, Alex J. Smola, and Stephane Canu. Heteroscedastic Gaussian Process regression. *Procedings of ICML-05*, pages 489–496, 2005.

[6] Edward Snelson and Zoubin Ghahramani. Variable noise and dimensionality reduction for sparse gaussian processes. *Procedings of UAI-06*, 2006.

[7] A.G. Wilson and Z. Ghahramani. Copula processes. In J. Lafferty, C. K. I. Williams, J. Shawe-Taylor, R.S. Zemel, and A. Culotta, editors, *Advances in Neural Information Processing Systems 23*, pages 2460–2468. 2010.

[8] Andrew Wilson and Zoubin Ghahramani. Generalised Wishart Processes. In *Proceedings of the Twenty-Seventh Conference Annual Conference on Uncertainty in Artificial Intelligence (UAI-11)*, pages 736–744, Corvallis, Oregon, 2011. AUAI Press.

[9] P. Dallaire, C. Besse, and B. Chaib-draa. Learning Gaussian Process Models from Uncertain Data. *16th International Conference on Neural Information Processing*, 2008.

[10] E. Solak, R. Murray-Smith, W.e. Leithead, D.J. Leith, and C.E. Rasmussen. Derivative observations in Gaussian Process models of dynamic systems. *NIPS-03*, pages 1033–1040, 2003.

[11] Agathe Girard, Carl Edward Rasmussen, Joaquin Quinonero Candela, and Roderick Murray-Smith. Gaussian Process priors with incertain inputs - application to multiple-step ahead time series forecasting. *Advances in Neural Information Processing Systems 16*, 2003.

